# A Fast Stochastic Error-Descent Algorithm for Supervised Learning and Optimization

Gert Cauwenberghs
California Institute of Technology
Mail-Code 128-95
Pasadena, CA 91125
E-mail: gert@cco.caltech.edu

## Abstract

A parallel stochastic algorithm is investigated for error-descent learning and optimization in deterministic networks of arbitrary topology. No *explicit* information about internal network structure is needed. The method is based on the model-free distributed learning mechanism of Dembo and Kailath. A modified parameter update rule is proposed by which each individual parameter vector perturbation contributes a decrease in error. A substantially faster learning speed is hence allowed. Furthermore, the modified algorithm supports learning time-varying features in dynamical networks. We analyze the convergence and scaling properties of the algorithm, and present simulation results for dynamic trajectory learning in recurrent networks.

## 1 Background and Motivation

We address general optimization tasks that require finding a set of constant parameter values $p_i$ that minimize a given error functional $\mathcal{E}(\mathbf{p})$. For supervised learning, the error functional consists of some quantitative measure of the deviation between a desired state $\mathbf{x}^T$ and the actual state of a network $\mathbf{x}$, resulting from an input $\mathbf{y}$ and the parameters $\mathbf{p}$. In such context the components of $\mathbf{p}$ consist of the connection strengths, thresholds and other adjustable parameters in the network. A

typical specification for the error in learning a discrete set of pattern associations $(\mathbf{y}^{(\alpha)}, \mathbf{x}^{T(\alpha)})$ for a steady-state network is the Mean Square Error (MSE)

$$\mathcal{E}(\mathbf{p}) = \frac{1}{2} \sum_{\alpha} \sum_{k} (x_k^{T(\alpha)} - x_k^{(\alpha)})^2 \tag{1}$$

and similarly, for learning a desired response $(\mathbf{y}(t), \mathbf{x}^T(t))$ in a dynamic network

$$\mathcal{E}(\mathbf{p}) = \frac{1}{2} \int_{t_0}^{t_f} \sum_{k} (x_k^T(t) - x_k(t))^2 \mathrm{d}t \quad . \tag{2}$$

For $\mathcal{E}(\mathbf{p})$ to be uniquely defined in the latter dynamic case, initial conditions $\mathbf{x}(t_{\mathrm{init}})$ need to be specified.

A popular method for minimizing the error functional is steepest error descent (gradient descent) [1]-[6]

$$\Delta \mathbf{p} = -\eta \frac{\partial \mathcal{E}}{\partial \mathbf{p}} \quad . \tag{3}$$

Iteration of (3) leads asymptotically to a local minimum of $\mathcal{E}(\mathbf{p})$, provided $\eta$ is strictly positive and small. The computation of the gradient is often cumbersome, especially for time-dependent problems [2]-[5], and is even ill-posed for analog hardware learning systems that unavoidably contain unknown process impurities. This calls for error descent methods avoiding *calculation* of the gradients but rather probing the dependence of the error on the parameters *directly*. Methods that use some degree of explicit internal information other than the adjustable parameters, such as Madaline III [6] which assumes a specific feedforward multi-perceptron network structure and requires access to internal nodes, are therefore excluded. Two typical methods which satisfy the above condition are illustrated below:

- **Weight Perturbation** [7], a simple sequential parameter perturbation technique. The method updates the individual parameters in sequence, by measuring the change in error resulting from a perturbation of a single parameter and adjusting that parameter accordingly. This technique effectively measures the components of the gradient sequentially, which for a complete knowledge of the gradient requires as many computation cycles as there are parameters in the system.

- **Model-Free Distributed Learning** [8], which is based on the "M.I.T." rule in adaptive control [9]. Inspired by analog hardware, the distributed algorithm makes use of time-varying perturbation signals $\pi_i(t)$ supplied in parallel to the parameters $p_i$, and correlates these $\pi_i(t)$ with the instantaneous network response $\mathcal{E}(\mathbf{p} + \pi)$ to form an incremental update $\Delta p_i$. Unfortunately, the distributed model-free algorithm does not support learning of dynamic features (2) in networks with delays, and the learning speed degrades sensibly with increasing number of parameters [8].

## 2   Stochastic Error-Descent: Formulation and Properties

The algorithm we investigate here combines both above methods, yielding a significant improvement in performance over both. Effectively, at *every* epoch the constructed algorithm decreases the error along a single randomly selected direction in the parameter space. Each such decrement is performed using a *single*

synchronous parallel parameter perturbation per epoch. Let $\hat{\mathbf{p}} = \mathbf{p} + \boldsymbol{\pi}$ with parallel perturbations $\pi_i$ selected from a random distribution. The perturbations $\pi_i$ are assumed reasonably small, but not necessarily mutually orthogonal. For a given *single* random instance of the perturbation $\boldsymbol{\pi}$, we update the parameters with the rule

$$\Delta\mathbf{p} = -\mu\,\hat{\mathcal{E}}\,\boldsymbol{\pi}\ , \tag{4}$$

where the scalar

$$\hat{\mathcal{E}} = \mathcal{E}(\hat{\mathbf{p}}) - \mathcal{E}(\mathbf{p}) \tag{5}$$

is the error contribution due to the perturbation $\boldsymbol{\pi}$, and $\mu$ is a small strictly positive constant. Obviously, for a sequential activation of the $\pi_i$, the algorithm reduces to the weight perturbation method [7]. On the other hand, by omitting $\mathcal{E}(\mathbf{p})$ in (5) the original distributed model-free method [8] is obtained. The subtraction of the unperturbed reference term $\mathcal{E}(\mathbf{p})$ in (5) contributes a significant increase in speed over the original method. Intuitively, the incremental error $\hat{\mathcal{E}}$ specified in (5) isolates the *specific* contribution due to the perturbation, which is obviously more relevant than the total error which includes a bias $\mathcal{E}(\mathbf{p})$ unrelated to the perturbation $\boldsymbol{\pi}$. This bias necessitates stringent zero-mean and orthogonality conditions on the $\pi_i$ and requires many perturbation cycles in order to effect a consistent decrease in the error [8].[1] An additional difference concerns the assumption on the dynamics of the perturbations $\pi_i$. By fixing the perturbation $\boldsymbol{\pi}$ during every epoch in the present method, the dynamics of the $\pi_i$ no longer interfere with the time delays of the network, and dynamic optimization tasks as (2) come within reach.

The rather simple and intuitive structure (4) and (5) of the algorithm is somewhat reminiscent of related models for reinforcement learning, and likely finds parallels in other fields as well. Random direction and line-search error-descent algorithms for trajectory learning have been suggested and analyzed by P. Baldi [12]. As a matter of coincidence, independent derivations of basically the same algorithm but from different approaches are presented in this volume as well [13],[14]. Rather than focussing on issues of originality, we proceed by analyzing the virtues and scaling properties of this method. We directly present the results below, and defer the formal derivations to the appendix.

2.1   The algorithm performs **gradient descent on average**, provided that the perturbations $\pi_i$ are mutually uncorrelated with uniform auto-variance, that is $\mathrm{E}(\pi_i\pi_j) = \sigma^2\delta_{ij}$ with $\sigma$ the perturbation strength. The effective gradient descent learning rate corresponding to (3) equals $\eta_{\mathrm{eff}} = \mu\sigma^2$.

Hence on average the learning trajectory follows the steepest path of error descent. The stochasticity of the parameter perturbations gives rise to fluctuations around the mean path of descent, injecting diffusion in the learning process. However, the individual fluctuations satisfy the following desirable regularity:

**2.2**   The error $\mathcal{E}(\mathbf{p})$ always decreases under an update (4) for *any* $\boldsymbol{\pi}$, provided that $|\boldsymbol{\pi}|^2$ is "small", and $\mu$ is strictly positive and "small".

Therefore, the algorithm is guaranteed to converge towards local error minima just like gradient descent, as long as the perturbation vector $\pi$ statistically explores all directions of the parameter space, provided the perturbation strength and learning rate are sufficiently small. This property holds only for methods which bypass the bias due to the offset error term $\mathcal{E}(\mathbf{p})$ for the calculation of the updates, as is performed here by subtraction of the offset in (5).

The guaranteed decrease in error of the update (4) under any small, single instance of the perturbation $\pi$ removes the need of averaging multiple trials obtained by different instances of $\pi$ in order to reduce turbulence in the learning dynamics. We intentionally omit any smoothing operation on the constructed increments (4) prior to effecting the updates $\Delta p_i$, unlike the estimation of the true gradient in [8],[10],[13] by essentially accumulating and averaging contributions (4) over a large set of random perturbations. Such averaging is unnecessary here (and in [13]) since each individual increment (4) contributes a decrease in error, and since the smoothing of the ragged downward trajectory on the error surface is effectively performed by the integration of the incremental updates (4) anyway. Furthermore, from a simple analysis it follows that such averaging is actually detrimental to the effective speed of convergence.[2]   For a correct measure of the convergence speed of the algorithm relative to that of other methods, we studied the boundaries of learning stability regions specifying maximum learning rates for the different methods. The analysis reveals the following scaling properties with respect to the size of the trained network, characterized by the number of adjustable parameters $P$:

**2.3**   The **maximum attainable average speed** of the algorithm is a factor $P^{1/2}$ slower than that of pure gradient descent, as opposed to the maximum average speed of sequential weight perturbation which is a factor $P$ slower than gradient descent.

The reduction in speed of the algorithm vs. gradient descent by the square root of the number of parameters can be understood as well from an information-theoretical point of view using physical arguments. At each epoch, the stochastic algorithm applies perturbations in all $P$ dimensions, injecting information in $P$ different "channels". However, only *scalar* information about the global response of the network to the perturbations is available at the outside, through a single "channel". On average, such an algorithm can extract knowledge about the response of the network in at most $P^{1/2}$ effective dimensions, where the upper limit is reached only if the perturbations are truly statistically independent, exploiting the full channel capacity. In the worst case the algorithm only retains scalar information through a single, low-bandwidth channel, which is e.g. the case for the sequential weight perturbation algorithm. Hence, the stochastic algorithm achieves a speed-up of a factor $P^{1/2}$ over the technique of sequential weight perturbation, by using parallel statistically independent perturbations as opposed to serial single perturbations. The original model-free algorithm by Dembo and Kailath [8] does not achieve this $P^{1/2}$

speed-up over the sequential perturbation method (and may even do worse), partly because the information about the specific error contribution by the perturbations is contaminated due to the constant error bias signal $\mathcal{E}(\mathbf{p})$.

Note that up to here the term "speed" was defined in terms of the number of epochs, which does not necessarily directly relate to the physical speed, in terms of the total number of operations. An equally important factor in speed is the amount of computation involved per epoch to obtain values for the updates (3) and (4). For the stochastic algorithm, the most intensive part of the computation involved at every epoch is the evaluation of $\mathcal{E}(\mathbf{p})$ for two instances of $\mathbf{p}$ in (5), which typically scales as $\mathcal{O}(P)$ for neural networks. The remaining operations relate to the generation of random perturbations $\pi_i$ and the calculation of the correlations in (4), scaling as $\mathcal{O}(P)$ as well. Hence, for an accurate comparison of the learning speed, the scaling of the computations involved in a single gradient descent step needs to be balanced against the computation effort by the stochastic method corresponding to an equivalent error descent rate, which combining both factors scales as $\mathcal{O}(P^{3/2})$. An example where the scaling for this computation balances in favor of the stochastic error-descent method, due to the expensive calculation of the full gradient, will be demonstrated below for dynamic trajectory learning.

More importantly, the intrinsic parallelism, fault tolerance and computational simplicity of the stochastic algorithm are especially attractive with hardware implementations in mind. The complexity of the computations can be furthermore reduced by picking a *binary* random distribution for the parallel perturbations, $\pi_i = \pm\sigma$ with equal probability for both polarities, simplifying the multiply operations in the parameter updates. In addition, powerful techniques exist to generate large-scale streams of pseudo-random bits in VLSI [15].

## 3   Numerical Simulations

For a test of the learning algorithm on time-dependent problems, we selected dynamic trajectory learning (a "Figure 8") as a representative example [2]. Several exact gradient methods based on an error functional of the form (2) exist [2]-[5], with a computational complexity scaling as either $\mathcal{O}(P)$ per epoch for an *off-line*[3] method [2] (requiring history storage over the complete time interval of the error functional), or as $\mathcal{O}(P^2)$ [3] and recently as $\mathcal{O}(P^{3/2})$ [4]-[5] per epoch for an *on-line* method (with only most current history storage). The stochastic error-descent algorithm provides an *on-line* alternative with an $\mathcal{O}(P)$ per epoch complexity. As a consequence, including the extra $P^{1/2}$ factor for the convergence speed relative to gradient descent, the overall computation complexity of the stochastic error-descent still scales like the best on-line exact gradient method currently available.

For the simulations, we compared several runs of the stochastic method with a single run of an exact gradient-descent method, all runs starting from the same initial conditions. For a meaningful comparison, the equivalent learning rate for

stochastic descent $\eta_{\text{eff}} = \mu\sigma^2$ was set to $\eta$, resulting in equal average speeds. We implemented binary random perturbations $\pi_i = \pm\sigma$ with $\sigma = 1 \times 10^{-3}$. We used the network topology, the teacher forcing mechanism, the values for the learning parameters and the values for the initial conditions from [4], case 4, except for $\eta$ (and $\eta_{\text{eff}}$) which we reduced from 0.1 to 0.05 to avoid strong instabilities in the stochastic sessions. Each epoch represents one complete period of the figure eight. We found no local minima for the learning problem, and all sessions converged successfully within 4000 epochs as shown in Fig. 1 (a). The occasional upward transitions in the stochastic error are caused by temporary instabilities due to the elevated value of the learning rate. At lower values of the learning rate, we observed significantly less frequent and articulate upward transitions. The measured distribution for the decrements in error at $\eta_{\text{eff}} = 0.01$ is given in Fig. 1 (b). The values of the stochastic error decrements in the histogram are normalized to the mean of the distribution, *i.e.* the error decrements by gradient descent (8). As expected, the error decreases at practically all times with an average rate equal to that of gradient descent, but the largest fraction of the updates cause little change in error.

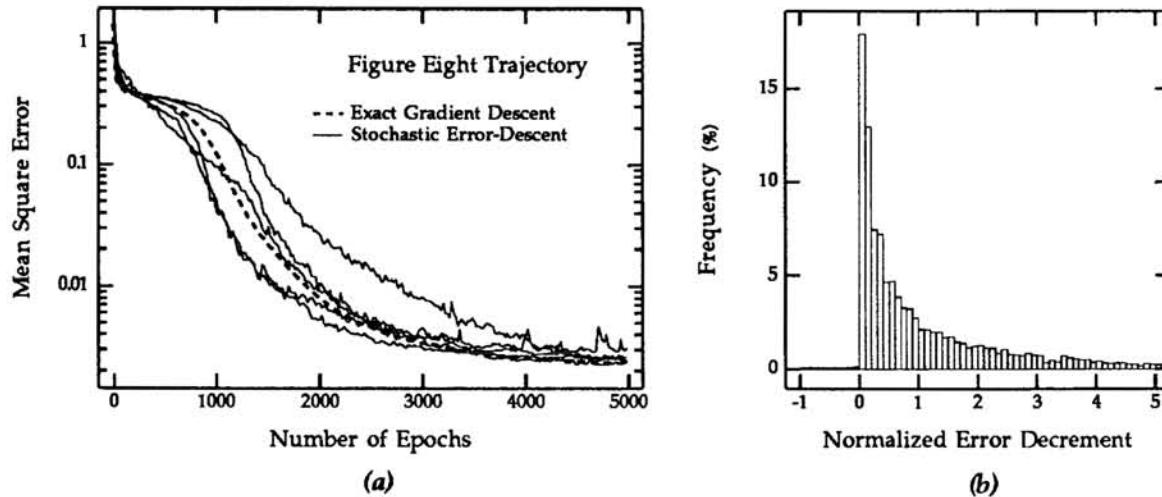

**Figure 1** Exact Gradient and Stochastic Error-Descent Methods for the Figure "8" Trajectory. (a) Convergence Dynamics ($\eta = 0.05$). (b) Distribution of the Error Decrements.($\eta = 0.01$).

## 4   Conclusion

The above analysis and examples serve to demonstrate the solid performance of the error-descent algorithm, in spite of its simplicity and the minimal requirements on explicit knowledge of internal structure. While the functional simplicity and fault-tolerance of the algorithm is particularly suited for hardware implementations, on conventional digital computers its efficiency compares favorably with pure gradient descent methods for certain classes of networks and optimization problems, owing to the involved effort to obtain full gradient information. The latter is particularly true for complex optimization problems, such as for trajectory learning and adaptive control, with expensive scaling properties for the calculation of the gradient. In particular, the discrete formulation of the learning dynamics, decoupled from the dynamics of the network, enables the stochastic error-descent algorithm to handle dynamic networks and time-dependent optimization functionals gracefully.

## Appendix: Formal Analysis

We analyze the algorithm for small perturbations $\pi_i$, by expanding (5) into a Taylor series around p:

$$\hat{\mathcal{E}} = \sum_{j} \frac{\partial \mathcal{E}}{\partial p_j} \pi_j + \mathcal{O}(|\pi|^2) \ , \tag{6}$$

where the $\partial \mathcal{E}/\partial p_j$ represent the components of the true error gradient, reflecting the physical structure of the network. Substituting (6) in (4) yields:

$$\Delta p_i = -\mu \sum_{j} \frac{\partial \mathcal{E}}{\partial p_j} \pi_i \pi_j + \mathcal{O}(|\pi|^2)\pi_i \ . \tag{7}$$

For mutually uncorrelated perturbations $\pi_i$ with uniform variance $\sigma^2$, $\mathrm{E}(\pi_i\pi_j) = \sigma^2\delta_{ij}$, the parameter vector on average changes as

$$\mathrm{E}(\Delta \mathrm{p}) = -\mu\sigma^2 \frac{\partial \mathcal{E}}{\partial \mathrm{p}} + \mathcal{O}(\sigma^3) \ . \tag{8}$$

Hence, on average the algorithm performs pure gradient descent as in (3), with an effective learning rate $\eta = \mu\sigma^2$. The fluctuations of the parameter updates (7) with respect to their average (8) give rise to diffusion in the error-descent process. Nevertheless, regardless of these fluctuations the error will *always* decrease under the updates (4), provided that the increments $\Delta p_i$ are sufficiently small ($\mu$ small):

$$\Delta \mathcal{E} = \sum_{i} \frac{\partial \mathcal{E}}{\partial p_i}\Delta p_i + \mathcal{O}(|\Delta \mathrm{p}|^2) \approx -\mu \sum_{i}\sum_{j} \frac{\partial \mathcal{E}}{\partial p_i}\pi_i \frac{\partial \mathcal{E}}{\partial p_j}\pi_j \approx -\mu\,\hat{\mathcal{E}}^2 \leq 0 \ . \tag{9}$$

Note that this is a direct consequence of the offset bias subtraction in (5), and (9) is no longer valid when the compensating reference term $\mathcal{E}(\mathrm{p})$ in (5) is omitted. The algorithm will converge towards local error minima just like gradient descent, as long as the perturbation vector $\pi$ statistically explores all directions of the parameter space. In principle, statistical independence of the $\pi_i$ is not required to ensure convergence, though in the case of cross-correlated perturbations the learning trajectory (7) does not on average follow the steepest path (8) towards the optima, resulting in slower learning.

The constant $\mu$ cannot be increased arbitrarily to boost the speed of learning. The value of $\mu$ is constrained by the allowable range for $|\Delta \mathrm{p}|$ in (9). The maximum level for $|\Delta \mathrm{p}|$ depends on the steepness and nonlinearity of the error functional $\mathcal{E}$, but is largely independent of which algorithm is being used. A value of $|\Delta \mathrm{p}|$ exceeding the limit will likely cause instability in the learning process, just as it would for an exact gradient descent method. The constraint on $|\Delta \mathrm{p}|$ allows us to formulate the maximum attainable speed of the stochastic algorithm, relative to that of other methods. From (4),

$$|\Delta \mathrm{p}|^2 = \mu^2|\pi|^2\hat{\mathcal{E}}^2 \approx P\mu^2\sigma^2\hat{\mathcal{E}}^2 \tag{10}$$

where $P$ is the number of parameters. The approximate equality at the end of (10) holds for large $P$, and results from the central limit theorem for $|\pi|^2$ with $\mathrm{E}(\pi_i\pi_j) = \sigma^2\delta_{ij}$. From (6), the expected value of (10) is

$$\mathrm{E}(|\Delta \mathrm{p}|^2) = P \left(\mu\sigma^2\right)^2 \left|\frac{\partial \mathcal{E}}{\partial \mathrm{p}}\right|^2 \ . \tag{11}$$

The maximum attainable value for $\mu$ can be expressed in terms of the maximum value of $\eta$ for gradient descent learning. Indeed, from a worst-case analysis of (3)

$$|\Delta \mathrm{p}|^2_{\max} = \eta^2_{\max} \left|\frac{\partial \mathcal{E}}{\partial \mathrm{p}}\right|^2_{\max} \tag{12}$$

and from a similar worst-case analysis of (11), we obtain $P\mu_{max}\sigma^2 \sim \eta_{max}$ to a first order approximation. With the derived value for $\mu_{max}$, the maximum effective learning rate $\eta_{eff}$ associated with the mean field equation (8) becomes $\eta_{eff} = P^{-1/2} \eta_{max}$ for the stochastic method, as opposed to $\eta_{max}$ for the exact gradient method. This implies that on average and under optimal conditions the learning process for the stochastic error descent method is a factor $P^{1/2}$ slower than optimal gradient descent. From similar arguments, it can be shown that for *sequential* perturbations $\pi_i$ the effective learning rate for the mean field gradient descent satisfies $\eta_{eff} = P^{-1} \eta_{max}$. Hence under optimal conditions the sequential weight perturbation technique is a factor $P$ slower than optimal gradient descent.

## Acknowledgements

We thank J. Alspector, P. Baldi, B. Flower, D. Kirk, M. van Putten, A. Yariv, and many other individuals for valuable suggestions and comments on the work presented here.

## Footnotes

[1] An interesting noise-injection variant on the model-free distributed learning paradigm of [8], presented in [10], avoids the bias due to the offset level $\mathcal{E}(\mathbf{p})$ as well, by differentiating the perturbation and error signals prior to correlating them to construct the parameter increments. A complete demonstration of an analog VLSI system based on this approach is presented in this volume [11]. As a matter of fact, the modified noise-injection algorithm corresponds to a continuous-time version of the algorithm presented here, for networks and error functionals free of time-varying features.

[2] Sure enough, averaging say $M$ instances of (4) for different random perturbations will improve the estimate of the gradient by decreasing its variance. However, the variance of the update $\Delta\mathbf{p}$ decreases by a factor of $M$, allowing an increase in learning rate by only a factor of $M^{1/2}$, while to that purpose $M$ network evaluations are required. In terms of total computation efforts, the averaged method is hence a factor $M^{1/2}$ slower.

[3]The distinction between on-line and off-line methods here refers to issues of time reversal in the computation. On-line methods process incoming data strictly in the order it is received, while off-line methods require extensive access to previously processed data. On-line methods are therefore more desirable for real-time learning applications.

## References

[1] D.E. Rumelhart, G.E. Hinton, and R.J. Williams, "Learning Internal Representations by Error Propagation," in *Parallel Distributed Processing, Explorations in the Microstructure of Cognition*, vol. 1, D.E. Rumelhart and J.L. McClelland, eds., Cambridge, MA: MIT Press, 1986.

[2] B.A. Pearlmutter, "Learning State Space Trajectories in Recurrent Neural Networks," *Neural Computation*, vol. 1 (2), pp 263-269, 1989.

[3] R.J. Williams and D. Zipser, "A Learning Algorithm for Continually Running Fully Recurrent Neural Networks," *Neural Computation*, vol. 1 (2), pp 270-280, 1989.

[4] N.B. Toomarian, and J. Barhen, "Learning a Trajectory using Adjoint Functions and Teacher Forcing," *Neural Networks*, vol. 5 (3), pp 473-484, 1992.

[5] J. Schmidhuber, "A Fixed Size Storage $\mathcal{O}(n^3)$ Time Complexity Learning Algorithm for Fully Recurrent Continually Running Networks," *Neural Computation*, vol. 4 (2), pp 243-248, 1992.

[6] B. Widrow and M.A. Lehr, "30 years of Adaptive Neural Networks. Perceptron, Madaline, and Backpropagation," *Proc. IEEE*, vol. 78 (9), pp 1415-1442, 1990.

[7] M. Jabri and B. Flower, "Weight Perturbation: An Optimal Architecture and Learning Technique for Analog VLSI Feedforward and Recurrent Multilayered Networks," *IEEE Trans. Neural Networks*, vol. 3 (1), pp 154-157, 1992.

[8] A. Dembo and T. Kailath, "Model-Free Distributed Learning," *IEEE Trans. Neural Networks*, vol. 1 (1), pp 58-70, 1990.

[9] H.P. Whitaker, "An Adaptive System for the Control of Aircraft and Spacecraft," in *Institute for Aeronautical Sciences*, pap. 59-100, 1959.

[10] B.P. Anderson and D.A. Kerns, "Using Noise Injection and Correlation in Analog Hardware to Estimate Gradients," *submitted*, 1992.

[11] D. Kirk, D. Kerns, K. Fleischer, and A. Barr, "Analog VLSI Implementation of Gradient Descent," in *Advances in Neural Information Processing Systems*, San Mateo, CA: Morgan Kaufman Publishers, vol. 5, 1993.

[12] P. Baldi, "Learning in Dynamical Systems: Gradient Descent, Random Descent and Modular Approaches," JPL Technical Report, California Institute of Technology, 1992.

[13] J. Alspector, R. Meir, B. Yuhas, and A. Jayakumar, "A Parallel Gradient Descent Method for Learning in Analog VLSI Neural Networks," in *Advances in Neural Information Processing Systems*, San Mateo, CA: Morgan Kaufman Publishers, vol. 5, 1993.

[14] B. Flower and M. Jabri, "Summed Weight Neuron Perturbation: An $\mathcal{O}(n)$ Improvement over Weight Perturbation," in *Advances in Neural Information Processing Systems*, San Mateo, CA: Morgan Kaufman Publishers, vol. 5, 1993.

[15] J. Alspector, J.W. Gannett, S. Haber, M.B. Parker, and R. Chu, "A VLSI-Efficient Technique for Generating Multiple Uncorrelated Noise Sources and Its Application to Stochastic Neural Networks," *IEEE T. Circuits and Systems*, 38 (1), pp 109-123, 1991.
